# Efficient high-dimensional maximum entropy modeling via symmetric partition functions

**Paul Vernaza**
The Robotics Institute
Carnegie Mellon University
Pittsburgh, PA 15213
pvernaza@cmu.edu

**J. Andrew Bagnell**
The Robotics Institute
Carnegie Mellon University
Pittsburgh, PA 15213
dbagnell@ri.cmu.edu

## Abstract

Maximum entropy (MaxEnt) modeling is a popular choice for sequence analysis in applications such as natural language processing, where the sequences are embedded in discrete, tractably-sized spaces. We consider the problem of applying MaxEnt to distributions over paths in *continuous spaces* of *high dimensionality*— a problem for which inference is generally intractable. Our main contribution is to show that this intractability can be avoided as long as the constrained features possess a certain kind of low dimensional structure. In this case, we show that the associated *partition function* is symmetric and that this symmetry can be exploited to compute the partition function efficiently in a compressed form. Empirical results are given showing an application of our method to learning models of high-dimensional human motion capture data.

## 1 Introduction

This work aims to generate useful probabilistic models of high dimensional trajectories in continuous spaces. This is illustrated in Fig. 1, which demonstrates the application of our proposed method to the problem of building generative models of high dimensional human motion capture data. Using this method, we may efficiently learn models and perform inferences including but not limited to the following: (1) Given any single pose, what is the probability that a certain type of motion ever visits this pose? (2) Given any pose, what is the distribution over future positions of the actor's hands? (3) Given any initial sequence of poses, what are the odds that this sequence corresponds to one action type versus another? (4) What is the most likely sequence of poses interpolating any two states?

The maximum entropy learning (MaxEnt) approach advocated here has the distinct advantage of being able to efficiently answer all of the aforementioned *global inferences* in a unified framework while also allowing the use of *global features* of the state and observations. In this sense, it is analogous to another MaxEnt learning method: the Conditional Random Field (CRF), which is typically applied to modeling discrete sequences. We show how MaxEnt modeling may be efficiently applied to paths in continuous state spaces of high dimensionality. This is achieved without having to resort to expensive, approximate inference methods based on MCMC, and without having to assume that the sequences themselves lie in or near a low dimensional submanifold, as in standard dimensionality-reduction-based methods. The key to our method is to make a natural assumption about the complexity of the *features*, rather than the paths, that results in simplifying symmetries.

This idea is illustrated in Fig. 2. Here we suppose that we are tasked with the problem of comparing two sets of paths: the first, sampled from an empirical distribution; and the second, sampled from a learned distribution intended to model the distribution underlying the empirical samples. Suppose first that we are to determine whether the learned distribution correctly samples the desired distribution. We claim that a natural approach to this problem is to visualize both sets of paths by projecting

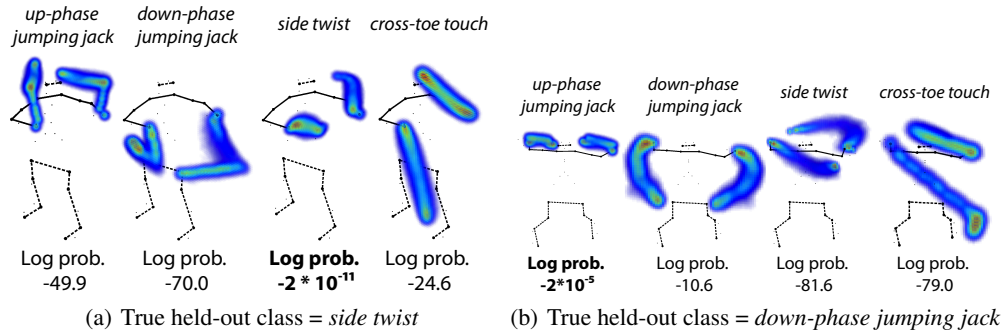

| up-phase jumping jack | down-phase jumping jack | side twist | cross-toe touch |
|---|---|---|---|
| Log prob. -49.9 | Log prob. -70.0 | **Log prob. -2 * 10⁻¹¹** | Log prob. -24.6 |

(a) True held-out class = *side twist*

Figure 1: Visualizations of predictions of future locations of hands for an individually held-out motion capture frame, conditioned on classes indicated by labels above figures, and corresponding class membership probabilities. See supplementary material for video demonstration.

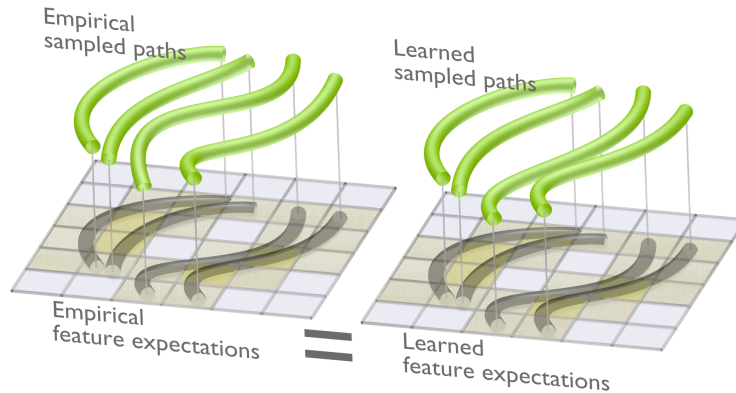

Figure 2: Illustration of the constraint that paths sampled from the learned distribution should (in expectation) visit certain regions of space exactly as often as they are visited by paths sampled from the true distribution, after projection of both onto a low dimensional subspace. The shading of each planar cell is proportional to the expected number of times that cell is visited by a path.

them onto a common low dimensional basis. If these projections appear similar, then we might conclude that the learned model is valid. If they do not appear similar, we might try to adjust the learned distribution, and compare projections again, iterating until the projections appear similar enough to convince us that the learned model is valid.

We then might consider automating this procedure by choosing numerical *features* of the projected paths and comparing these features in order to determine whether the projected paths appear similar. Our approach may be thought of as a way of formalizing this procedure. The MaxEnt method described here iteratively samples paths, projects them onto a low dimensional subspace, computes features of these projected paths, and adjusts the distribution so as to ensure that, in expectation, these features match the desired features.

A key contribution of this work is to show that that employing *low dimensional features* of this sort enables tractable inference and learning algorithms, even in high dimensional spaces. Maximum entropy learning requires repeatedly calculating feature statistics for different distributions, which generally requires computing average feature values over *all* paths sampled from the distributions. Though this is straightforward to accomplish via dynamic programming in low dimensional spaces, it may not be obvious that the same can be accomplished in high-dimensional spaces. We will show how this is possible by exploiting symmetries that result from this assumption.

The organization of this paper is as follows. We first review some preliminary material. We then continue with a detailed exposition of our method, followed by experimental results. Finally, we describe the relation of our method to existing methods and discuss conclusions.

## 2 Preliminaries

We now briefly review the basic MaxEnt modeling problem in discrete state spaces. In the basic MaxEnt problem, we have $N$ disjoint *events* $x_i$, $K$ random variables denoted *features* $\phi_j(x_i)$ mapping events to scalars, and $K$ expected values of these features $\mathbb{E}\phi_j$. To continue the example previously discussed, we will think of each $x_i$ as being a path, $\phi_j(x_i)$ as being the number of times that a path passes through the $j$th spatial region, and $\mathbb{E}\phi_j$ as the empirically estimated number of times that a path visits the $j$th region.

Our goal is to find a distribution $p(x_i)$ over the events consistent with our empirical observations in the sense that it generates the observed feature expectations:

$$\sum_i \phi_j(x_i)p(x_i) = \mathbb{E}\phi_j, \ \forall j \in \{1 \ldots K\}.$$

Of all such distributions, we will seek the one whose entropy is maximal [6]. This problem can be written compactly as

$$\max_{p \in \Delta} - \sum_i p_i \log p_i \text{ s.t. } \Phi p = \mathbb{E}\phi, \tag{1}$$

where we have defined vectors $p_i = p(x_i)$ and $\phi$, the feature matrix $\Phi_{ij} = \phi_i(x_j)$, and the probability simplex $\Delta$. Introducing a vector of Lagrange multipliers $\theta$, the Lagrangian dual of this concave maximization problem is [3]

$$\max_{\theta} - \log \left( \sum_i \exp(-\sum_j \Phi_{ji}\theta_j) \right) - \mathbb{E}\phi^T \theta. \tag{2}$$

It is straightforward to show that the gradient of the dual objective $g(\theta)$ is given by $\nabla_\theta g = \mathbb{E}_{\bar{p}}[\phi \mid \theta] - \mathbb{E}\phi$, where $\bar{p}$ is the *Gibbs* distribution over $x$ defined by

$$\bar{p}(x_i \mid \theta) \propto \exp \left( -\sum_j \phi_j(x_i)\theta_j \right). \tag{3}$$

## 3 MaxEnt modeling of continuous paths

We now consider an extension of the MaxEnt formalism to the case that the events are paths embedded in a continuous space. The main questions to be addressed here are how to handle the transition from a finite number of events to an infinite number of events, and how to define appropriate features. We will address the latter problem first.

We suppose that each event $x$ now consists of a continuous, arc-length-parameterized path, expressed as a function $\mathbb{R}^+ \to \mathbb{R}^N$ mapping a non-negative time into the *state space* $\mathbb{R}^N$. A natural choice in this case is to express each feature $\phi_j$ as an integral of the following form:

$$\phi_j(x) = \int_0^T \psi_j(x(s))ds, \tag{4}$$

where $T$ is the duration (or length) of $x$ and each $\psi_j : \mathbb{R}^N \to \mathbb{R}^+$ is what we refer to as a *feature potential*. Continuing the previous example, if we choose $\psi_j(x(t)) = 1$ if $x(t)$ is in region $j$ and $\psi_j(x(t)) = 0$ otherwise, then $\psi_j(x)$ is the total time that $x$ spends within the $j$th region of space.

An analogous expression for the probability of a continuous path is then obtained by substituting these features into (3). Defining the *cost function* $C_\theta := \sum_j \theta_j \psi_j$ and the *cost functional*

$$S_\theta\{x\} := \int_0^T C_\theta(x(s))ds, \tag{5}$$

we have that

$$\bar{p}(x \mid \theta) = \frac{\exp -S_\theta\{x\}}{\int \exp -S_\theta\{x\}\mathcal{D}x}, \tag{6}$$

where the notation $\int \exp -S_\theta \{x\} \mathcal{D}x$ denotes the integral of the cost functional over the space of all continuous paths. The normalization factor $Z_\theta := \int \exp -S_\theta \{x\} \mathcal{D}x$ is referred to as the *partition function*. As in the discrete case, computing the partition function is of prime concern, as it enables a variety of inference and learning techniques.

The *functional integral* in (6) can be formalized in several ways, including taking an expectation with respect to Wiener measure [12] or as a Feynman integral [4]. Computationally, evaluating $Z_\theta$ requires the solution of an elliptic partial differential equation over the state space, which can be derived via the Feynman-Kac theorem [12, 5]. The solution, denoted $Z_\theta(a)$ for $a \in \mathbb{R}^N$, gives the value of the functional integral evaluated over all paths beginning at $a$ and ending at a given goal location (henceforth assumed w.l.o.g. to be the origin).

A discrete approximation to the partition function can therefore be computed via standard numerical methods such as finite differences, finite elements, or spectral methods [2]. However, we proceed by discretizing the state space as a lattice graph and computing the partition function associated with discrete paths in this graph via a standard dynamic programming method [1, 15, 11]. Recent work has shown that this method recovers the PDE solution in the discretization limit [5]. Concretely, the discretized partition function is computed as the fixed point of the following iteration:

$$Z_\theta(a) \leftarrow \delta(a) + \exp(-\epsilon C_\theta(a)) \sum_{a' \sim a} Z_\theta(a'), \qquad (7)$$

where $a' \sim a$ denotes the set of $a'$ adjacent to $a$ in the lattice, $\epsilon$ is the spacing between adjacent lattice elements, and $\delta$ is the Kronecker delta. [1]

## 4 Efficient inference via symmetry reduction

Unfortunately, the dynamic programming approach described above is tractable only for low dimensional problems; for problems in more than a few dimensions, even storing the partition function would be infeasible. Fortunately, we show in this section that it is possible to compute the partition function directly in a compressed form, given that the features also satisfy a certain compressibility property.

### 4.1 Symmetry of the partition function

Elaborating on this statement, we now recall Eq. (4), which expresses the features as integrals of feature potentials $\psi_j$ over paths. We then examine the effects of assuming that the $\psi_j$ are compressible in the sense that they may be predicted exactly from their projection onto a low dimensional subspace—i.e., we assume that

$$\psi_j(a) = \psi_j(WW^T a), \; \forall j, a, \qquad (8)$$

for some given $N \times d$ matrix $W$, with $d < N$. The following results show that compressibility of the features in this sense implies that the corresponding partition function is also compressible, in the sense that we need only compute it restricted to a $d+1$ dimensional subspace in order to determine its values at arbitrary locations in $N$-dimensional space. This is shown in two steps. First, we show that the partition function is symmetric about rotations about the origin that preserve the subspace spanned by the columns of $W$. We then show that there always exists such a rotation that also brings an arbitrary point in $\mathbb{R}^N$ into correspondence with a point in a a $d+1$-dimensional *slice* where the partition function has been computed.

**Theorem 4.1.** *Let $Z_\theta = \int \exp -S_\theta \{x\} \mathcal{D}x$, with $S_\theta$ as defined in Eq. 5 and features derived from feature potentials $\psi_j$. Suppose that $\psi_j(x) = \psi_j(WW^T x), \; \forall j, x$. Then for any orthogonal $R$ such that $RW = W$,*

$$Z_\theta(a) = Z_\theta(Ra), \; \forall a \in \mathbb{R}^N. \qquad (9)$$

*Proof.* By definition,

$$Z_\theta(Ra) = \int_{\substack{x(0)=0 \\ x(T)=Ra}} \exp \left( -\int_0^T C_\theta(x(s)) ds \right) \mathcal{D}x.$$

The substitution $y(t) = R^T x(t)$ yields

$$Z_\theta(Ra) = \int_{\substack{y(0)=0 \\ y(T)=a}} \exp\left(-\int_0^T C_\theta(Ry(s))ds\right) \mathcal{D}y.$$

Since $\psi_j(a) = \psi_j(WW^T a)$, $\forall j, a$ implies that $C_\theta(x) = C_\theta(WW^T x) \, \forall x$, we can make the substitutions $C_\theta(Ry) = C_\theta(WW^T Ry) = C_\theta(WW^T y) = C_\theta(y)$ in the previous expression to prove the result. $\qquad\square$

The next theorem makes explicit how to exploit the symmetry of the partition function by computing it restricted to a low-dimensional *slice* of the state space.

**Corollary 4.2.** *Let $W$ be a matrix such that $\psi_j(a) = \psi_j(WW^T a)$, $\forall j, a$, and let $\nu$ be any vector such that $W^T \nu = 0$ and $\|\nu\| = 1$. Then*

$$Z_\theta(a) = Z_\theta(WW^T a + \|(I - WW^T)a\|\nu), \forall a \tag{10}$$

*Proof.* The proof of this result is to show that there always exists a rotation satisfying the conditions of Theorem 4.1 that rotates $b$ onto the subspace spanned by the columns of $W$ and $\nu$. We simply choose an $R$ such that $RW = W$ and $R(I - WW^T b) = \|I - WW^T b\|\nu$. That this is a valid rotation follows from the orthogonality of $W$ and $\nu$ and the unit-norm assumption on $\nu$. Applying any such rotation to $b$ proves the result. $\qquad\square$

## 4.2 Exploiting symmetry in DP

We proceed to compute the discretized partition function via a modified version of the dynamic programming algorithm described in Sec. 3. The only substantial change is that we leverage Corollary 4.2 in order to represent the partition function in a compressed form. This implies corresponding changes in the updates, as these must now be derived from the new, compressed representation.

Figure 3 illustrates the algorithm applied to computing the partition function associated with a constant $C(x)$ in a two-dimensional space. The partition function is represented by its values on a regular lattice lying in the low-dimensional *slice* spanned by the columns of $W$ and $\nu$, as defined in Corollary 4.2. In the illustrated example, $W$ is empty, and $\nu$ is any arbitrary line. At each iteration of the algorithm, we update each value in the slice based on adjacent values, as before. However, it is now the case that some of the adjacent nodes lie off of the slice. We compute the values associated with such nodes by rotating them onto the slice (according to Corollary 4.2) and interpolating the value based on those of adjacent nodes within the slice.

An explicit formula for these updates is readily obtained. Suppose that $b$ is a point contained within the slice and $y := b + \delta$ is an adjacent point lying off the slice whose value we wish to compute. By assumption, $W^T \delta = \nu^T \delta = 0$. We therefore observe that $\delta^T(I - WW^T)b = 0$, since $(I - WW^T)b \propto \nu$. Hence,

$$\begin{aligned} V(y) &= V(WW^T(b + \delta) + \|(I - WW^T)(b + \delta)\|\nu) \\ &= V(WW^T b + \|(I - WW^T)b + \delta\|\nu) \\ &= V(WW^T b + \sqrt{\|(I - WW^T)b\|^2 + \|\delta\|^2}\nu). \end{aligned} \tag{11}$$

An interesting observation is that this formula depends on $y$ only through $\|\delta\|$. Therefore, assuming that all nodes adjacent to $b$ lie at a distance of $\delta$ from it, all of the updates from the off-slice neighbors will be identical, which allows us to compute the net contribution due to all such nodes simply by multiplying the above value by their cardinality. The computational complexity of the algorithm is in this case independent of the dimension of the ambient space.

A detailed description of the algorithm is given in Algorithm 1.

## 4.3 MaxEnt training procedure

Given the ability to efficiently compute the partition function, learning may proceed in a way exactly analogous to the discrete case (Sec. 2). A particular complication in our case is that exactly

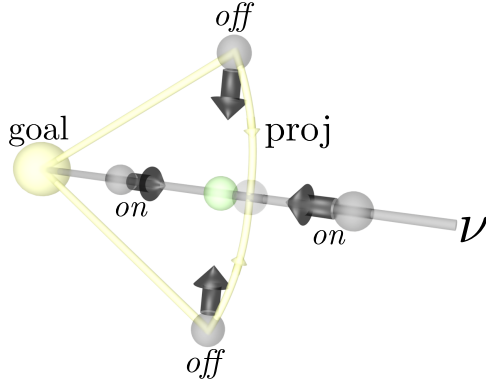

Figure 3: Illustration of dynamic programming update (constant cost example). The large sphere marked *goal* denotes origin with respect to which partition function is computed. Partition function in this case is symmetric about all rotations around the origin; hence, any value can be computed by rotation onto any axis (*slice*) where the partition function is known ($\nu$). Contributions from *off-slice* and *on-slice* points are denoted by *off* and *on*, respectively. Symmetry implies that value updates from *off-axis* nodes can be computed by rotation (proj) onto the axis. See supplementary material for video demonstration.

computing feature expectations under the model distribution is not as straightforward as in the low dimensional case, as we must account for the symmetry of the partition function. As such, we compute feature expectations by sampling paths from the model given the partition function.

---

**Algorithm 1** PartitionFunc($x_T, C_\theta, W, N, d$)

$Z : \mathbb{R}^{d+1} \to \mathbb{R} : y \mapsto 0$     {initialize partition function to zero}
$\nu \leftarrow (\nu \mid \langle \nu, \nu \rangle = 1, W^T \nu = 0)$     {choose an appropriate $\nu$}
lift : $\mathbb{R}^{d+1} \to \mathbb{R}^N : y \mapsto [W \; \nu] y + x_T$     {define lifting and projection operators}
proj : $\mathbb{R}^N \to \mathbb{R}^{d+1} : x \mapsto \begin{pmatrix} W^T(x - x_T) \\ \|(I - WW^T)(x - x_T)\| \end{pmatrix}$

**while** $Z$ not converged **do**
   **for** $y \in G \subset \mathbb{Z}^{d+1}$ **do**
      $z_{\text{on}} \leftarrow \sum_{\{\delta \in \mathbb{Z}^{d+1} \mid \|\delta\| = 1\}} Z(y' + \delta)$     {calculate *on-slice* contributions}
      $z_{\text{off}} \leftarrow 2(N - d - 1) Z(y_1, \dots, y_d, \sqrt{y_{d+1}^2 + 1})$     {calculate *off-slice* contributions}
      $Z(y) \leftarrow \frac{z_{\text{on}} + z_{\text{off}} + 2N\delta(y)}{2N(\exp \epsilon C_\theta(\text{lift}(y)))}$     {iterate fixed-point equation}
   **end for**
**end while**
$Z' : \mathbb{R}^N \to \mathbb{R} : x \mapsto Z(\text{proj}(x))$     {return partition function in original coordinates}
**return** $Z'$

---

## 5  Results

We implemented the method and applied it to the problem of modeling high dimensional motion capture data, as described in the introduction. Our training set consisted of a small sample of trajectories representing four different exercises performed by a human actor. Each sequence is represented as a 123-dimensional time series representing the Cartesian coordinates of 41 reflective markers located on the actor's body.

The feature potentials employed consisted of indicator functions of the form

$$\phi_j(a) = \{1 \text{ if } W^T a \in \mathcal{C}_j, \; 0 \text{ otherwise}\}, \tag{12}$$

where the $\mathcal{C}_j$ were non-overlapping, rectangular regions of the projected state space. A $W$ was chosen with two columns, using the method proposed in [13], which is effectively similar to performing PCA on the velocities of the trajectory.

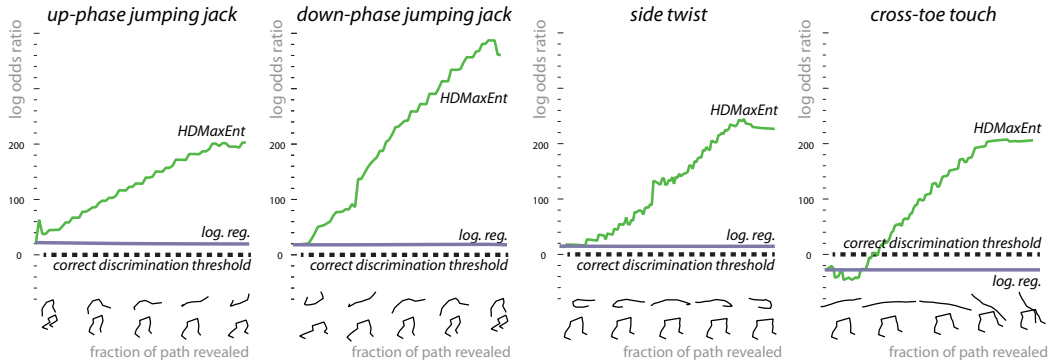

Figure 4: Results of classification experiment given progressively revealed trajectories. Title indicates true class of held-out trajectory. Abscissa indicates the fraction of the trajectory revealed to the classifiers. Samples of held-out trajectory at different points along abscissa are illustrated above *fraction of path revealed*. Ordinate shows predicted log-odds ratio between correct class and next-most-probable class.

We applied our method to train a maximum entropy model independently for each of the four classes. Given our ability to efficiently compute the partition function, this enables us to normalize each of these probability distributions. Classification can then be performed simply by evaluating the probability of a held-out example under each of the class models. Knowing the partition function also enables us to perform various marginalizations of the distribution that would otherwise be intractable. [8, 15]

In particular, we performed an experiment consisting of evaluating the probability of a held-out trajectory under each model as it was progressively revealed in time. This can be accomplished by evaluating the following quantity:

$$P(x_0)\gamma^t \exp\left(-\sum_{i=1}^{t}\epsilon C_\theta(x_i)\right)\frac{Z_\theta(x_t)}{Z_\theta(x_0)}, \tag{13}$$

where $x_0, \ldots, x_t$ represents the portion of the trajectory revealed up to time $t$, $P(x_0)$ is the prior probability of the initial state, and $\epsilon$ is the spacing between successive samples. Results of this experiment are shown in Fig. 4, which plots the predicted log-odds ratio between the correct and next-most-probable classes.

For comparison, we also implemented a classifier based on logistic regression. Features for this classifier consisted of radial basis functions centered around the portion of each training trajectory revealed up to the current time step. Both methods also employed the same prior initial state probability $P(x_0)$, which was constructed as a single isotropic Gaussian distribution for each class. Both classifiers therefore predict the same class distributions at time $t = 0$.

In the first three held-out examples, the initial state was distinctive enough to unambiguously predict the sequence label. The logistic regression predictions were generally inaccurate on their own, but the the confidence of these predictions was so low that these probabilities were far outweighed by the prior—the log-odds ratio in time therefore appears almost flat for logistic regression. Our method (denoted *HDMaxEnt* in the figure), on the other hand, demonstrated exponentially increasing confidence as the sequences were progressively revealed.

In the last example, the initial state appeared more similar to that of another class, causing the prior to mispredict its label. Logistic regression again exhibited no deviation from the prior in time. Our method, however, quickly recovered the correct label as the rest of the sequence was revealed.

Figures 1(a) and 1(b) show the result of a different inference—here we used the same learned class models to evaluate the probability that a *single* held-out frame was generated by a path in each class. This probability can be computed as the product of *forward* and *backwards* partition functions evaluated at the held-out frame divided by the partition function between nominal start and goal positions. [15] We also sampled trajectories given each potential class label, given the held-out frame as a starting point, and visualized the results.

The first held-out frame, displayed in Fig. 1(a), is distinctive enough that its marginal probability under the correct class, is far greater than its probability under any other class. The visualizations make it apparent that it is highly unlikely that this frame was sampled from one of the *jumping jack* paths, as this would require an unnatural excursion from the kinds of trajectory normally produced by those classes, while it is slightly more plausible that the frame could have been taken from a path sampled from the *cross-toe touch* class.

Fig. 1(b) shows a case where the held-out frame is ambiguous enough that it could have been generated by either the jumping jack *up* or *down* phases. In this case, the most likely prediction is incorrect, but it is still the case that the probabilities of the two plausible classes far outweigh those of the visibly less-plausible classes.

## 6 Related work

Our work bears the most relation to the extensive literature on maximum entropy modeling in sequence analysis. A well-known example of such a technique is the Conditional Random Field [9], which is applicable to modeling discrete sequences, such as those encountered in natural language processing. Our method is also an instance of MaxEnt modeling applied to sequence analysis; however, our method applies to high-dimensional paths in continuous spaces with a continuous notion of (potentially unbounded) time (as opposed to the discrete notions of finite sequence length or horizon). These considerations necessitate the development of the formulation and inference techniques described here.

Also notable are latent variable models that employ Gaussian process regression to probabilistically represent observation models and the latent dynamics [14, 10, 7]. Our method differs from these principally in two ways. First our method is able to exploit global, contextual features of sequences without having to model how these features are generated from a latent state. Although the features used in the experiments shown here were fairly simple, we plan to show in future work how our method can leverage context-dependent features to generalize across different environments. Second, global inferences in the aforementioned GP-based methods are intractable, since the state distribution as a function of time is generally not a Gaussian process, unless the dynamics are assumed linear. Therefore, expensive, approximate inference methods such as MCMC would be required to compute any of the inferences demonstrated here.

## 7 Conclusions

We have demonstrated a method for efficiently performing inference and learning for maximum-entropy modeling of high dimensional, continuous trajectories. Key to the method is the assumption that features arise from potentials that vary only in low dimensional subspaces. The partition functions associated with such features can be computed efficiently by exploiting the symmetries that arise in this case. The ability to efficiently compute the partition function enables tractable learning as well as the opportunity to compute a variety of inferences that would otherwise be intractable. We have demonstrated experimentally that the method is able to build plausible models of high dimensional motion capture trajectories that are well-suited for classification and other prediction tasks.

As future work, we would like to explore similar ideas to leverage more generic types of low dimensional structure that might arise in maximum entropy modeling. In particular, we anticipate that the method described here might be leveraged as a subroutine in future approximate inference methods for this class of problems. We are also investigating problem domains such as assistive teleoperation, where the ability to leverage contextual features is essential to learning policies that generalize.

## 8 Acknowledgments

This work is supported by the ONR MURI grant N00014-09-1-1052, Distributed Reasoning in Reduced Information Spaces.

## Footnotes

[1] In practice, this is typically done with respect to $\log Z_\theta$, which yields an iteration similar to a soft version of value iteration of the Bellman equation [15]

# References

[1] T. Akamatsu. Cyclic flows, markov process and stochastic traffic assignment. *Transportation Research Part B: Methodological*, 30(5):369–386, 1996.

[2] J.P. Boyd. *Chebyshev and Fourier spectral methods*. Dover, 2001.

[3] S.P. Boyd and L. Vandenberghe. *Convex optimization*. Cambridge Univ Pr, 2004.

[4] R.P. Feynman, A.R. Hibbs, and D.F. Styer. *Quantum Mechanics and Path Integrals: Emended Edition*. Dover Publications, 2010.

[5] S. García-Díez, E. Vandenbussche, and M. Saerens. A continuous-state version of discrete randomized shortest-paths, with application to path planning. In *CDC and ECC*, 2011.

[6] E.T. Jaynes. Information theory and statistical mechanics. *The Physical Review*, 106(4):620–630, 1957.

[7] J. Ko and D. Fox. Gp-BayesFilters: Bayesian filtering using Gaussian process prediction and observation models. *Autonomous Robots*, 27(1):75–90, 2009.

[8] D. Koller and N. Friedman. *Probabilistic Graphical Models: Principles and Techniques*. MIT Press, 2009.

[9] J. Lafferty. Conditional random fields: Probabilistic models for segmenting and labeling sequence data. In *ICML*, 2001.

[10] N.D. Lawrence and J. Quiñonero-Candela. Local distance preservation in the GP-LVM through back constraints. In *Proceedings of the 23rd international conference on Machine learning*, pages 513–520. ACM, 2006.

[11] A. Mantrach, L. Yen, J. Callut, K. Francoisse, M. Shimbo, and M. Saerens. The sum-over-paths covariance kernel: A novel covariance measure between nodes of a directed graph. *PAMI*, 32(6):1112–1126, 2010.

[12] B.K. Øksendal. *Stochastic differential equations: an introduction with applications*. Springer Verlag, 2003.

[13] P. Vernaza, D.D. Lee, and S.J. Yi. Learning and planning high-dimensional physical trajectories via structured lagrangians. In *ICRA*, pages 846–852. IEEE, 2010.

[14] J. Wang, D. Fleet, and A. Hertzmann. Gaussian process dynamical models. *NIPS*, 18:1441, 2006.

[15] Brian D. Ziebart, Andrew Maas, J. Andrew Bagnell, and Anind K. Dey. Maximum entropy inverse reinforcement learning. In *AAAI*, pages 1433–1438, 2008.

